# Optimising Synchronisation Times for Mobile Devices

**Neil D. Lawrence**
Department of Computer Science,
Regent Court, 211 Portobello Road,
Sheffield, S1 4DP, U.K.
neil@dcs.shef.ac.uk

**Antony I. T. Rowstron  Christopher M. Bishop  Michael J. Taylor**
Microsoft Research
7 J. J. Thomson Avenue
Cambridge, CB3 0FB, U.K.
{antr,cmbishop,mitaylor}@microsoft.com

## Abstract

With the increasing number of users of mobile computing devices (e.g. personal digital assistants) and the advent of third generation mobile phones, wireless communications are becoming increasingly important. Many applications rely on the device maintaining a *replica* of a data-structure which is stored on a server, for example news databases, calendars and e-mail. In this paper we explore the question of the optimal strategy for synchronising such replicas. We utilise probabilistic models to represent how the data-structures evolve and to model user behaviour. We then formulate objective functions which can be minimised with respect to the synchronisation timings. We demonstrate, using two real world data-sets, that a user can obtain more up-to-date information using our approach.

## 1 Introduction

As the available bandwidth for wireless devices increases, new challenges are presented in the utilisation of such bandwidth. Given that always up connections are generally considered infeasible an important area of research in mobile devices is the development of intelligent strategies for communicating between mobile devices and servers. In this paper we consider the scenario where we are interested in maintaining, on a personal digital assistant (PDA) with wireless access, an up-to-date *replica* of some, perhaps disparate, data-structures which are evolving in time. The objective is to make sure our replica is not 'stale'. We will consider a limited number of connections or *synchronisations*. Each synchronisation involves a *reconciliation* between the replica on the mobile device and the data-structures of interest on the server. Later in the paper we shall examine two typical examples of such an application, an internet news database and a user's e-mail messages. Currently the typical strategy[1] for performing such reconciliations is to synchronise every $M$ minutes,

where $M$ is a constant, we will call this strategy the *uniformly-spaced strategy*. We will make the timings of the synchronisations adaptable by developing a cost function that can be optimised with respect to the timings, thereby improving system performance.

## 2  Cost Function

We wish to minimise the *staleness* of the replica, where we define staleness as the time between an update of a portion of the data-structure on the server and the time of the synchronisation of that update with the PDA. For simplicity we shall assume that each time the PDA synchronises all the outstanding updates are transferred. Thus, after synchronisation the replica on the mobile device is consistent with the master copy on the server. Therefore, if $s_k$ is the time of the $k$th synchronisation in a day, and updates to the data-structure occur at times $u_j$ then the average staleness of the updates transferred during synchronisation $s_k$ will be

$$S_k = \sum_{j \text{ for } s_{k-1} < u_j \leq s_k} (s_k - u_j). \tag{1}$$

As well as staleness, we may be interested in optimising other criteria. For example, mobile phone companies may seek to equalise demand across the network by introducing time varying costs for the synchronisations, $c(t)$. Additionally one could argue that there is little point in keeping the replica fresh during periods when the user is unlikely to check his PDA, for example when he or she is sleeping. We might therefore want to minimise the time between the user's examination of the PDA and the last synchronisation. If the user looks at the PDA at times $a_i$ then we can express this as

$$F_k = \sum_{j \text{ for } s_{k-1} < a_i \leq s_k} (a_i - s_k). \tag{2}$$

Given the timings $u_j$ and $a_i$, the call cost schedule $c(t)$ and $K$ synchronisations, the total cost function may now be written

$$C = \sum_{k=1}^{K} \left( -\alpha F_k + \beta S_k + c(s_k) \right), \tag{3}$$

where $\alpha$ and $\beta$ are constants with units of $\frac{\text{money}}{\text{time}}$ which express the relative importance of the separate parts of the cost function. Unfortunately, of course, whilst we are likely to have knowledge of the call cost schedule, $c(t)$, we won't know the true timings $\{u_j\}$ and $\{a_i\}$ and the cost function will be *a priori* incomputable. If, though, we have historic data[2] relating to these times, we can seek to make progress by modelling these timings probabilistically. Then, rather than minimising the actual cost function, we can look to minimise the expectation of the cost function under these probabilistic models.

## 3  Expected Cost

There are several different possibilities for our modelling strategy. A sensible assumption is that there is independence between different parts of the data-structure (i.e. e-mail and business news can be modelled separately), however, there may be dependencies between update times which occur within the same part. The

periodicity of the data may be something we can take advantage of, but any non-stationarity in the data may cause problems. There are various model classes we could consider; for this work however, we restrict ourselves to stationary models, and ones in which updates arrive independently and in a periodic fashion.

Let us take $T$ to be the largest period of oscillation in the data arrivals, for a particular portion of a data-structure. We model this portion with a probability distribution, $p_u(t)$. Naturally more than one update may occur in that interval, therefore our probability distribution really specifies a distribution over time given one that one update (or user access) has occurred. To fully specify the model we also are required to store the expected number of updates, $J_u$, (or accesses, $J_a$) that occur in that interval.

The expected value of $S_k$ may now be written,

$$\langle S_k \rangle_{p_u(t)} = \int_{s_{k-1}}^{s_k} \lambda_u(t)(s_k - t)dt, \tag{4}$$

where $\langle \cdot \rangle_{p(x)}$ is an expectation under the distribution $p(x)$, $\lambda_u(t) = J_u p_u(t)$ can be viewed as the rate at which updates are occurring and $s_0 = s_K - T$.

We can model the user access times, $a_i$, in a similar manner, which leads us to the expected value of the freshness, $\langle F_k \rangle_{p_a(t)} = \int_{s_k}^{s_{k+1}} \lambda_a(t)(t - s_k)dt$, where $\lambda_a(t) = J_a p_a(t)$ The overall expected cost, which we will utilise as our objective function, may therefore be written

$$\langle C \rangle = \sum_{k=1}^{K} \left( \langle S_k \rangle_{p_u} - \langle F_k \rangle_{p_a} + c(s_k) \right). \tag{5}$$

### 3.1 Probabilistic Models.

We now have an objective function which is a function of the variables we wish to optimise, the synchronisation times, but whilst we have mentioned some characteristics of the models $p_u(t)$ and $p_a(t)$ we have not yet fully specified their form.

We have decreed that the models should be periodic and that they may consider each datum to occur independently. In effect we are modelling data which is mapped to a circle. Various options are available for handling such models; for this work, we constrain our investigations to kernel density estimates (KDE).

In order to maintain periodicity, we must select a basis function for our KDE which represents a distribution on a circle, one simple way of achieving this aim is to wrap a distribution that is defined along a line to the circle (Mardia, 1972). A traditional density which represents a distribution on the line, $p(t)$, may be wrapped around a circle of circumference $T$ to give us a distribution defined on the circle, $p(\theta)$, where $\theta = t \mod T$. This means a basis function with its centre at $T - \delta$, that will typically have probability mass when $u > T$, wraps around to maintain a continuous density at $T$. The wrapped Gaussian distribution[3] that we make use of takes the form

$$\mathcal{WN}(\theta|\mu, \sigma^2) = \frac{1}{\sqrt{2\pi\sigma^2}} \sum_{k=-\infty}^{\infty} \exp\left[ \frac{-(\theta - \mu + Tk)^2}{2\sigma^2} \right]. \tag{6}$$

The final kernel density estimate thus consists of mapping the data points $t_n \to \theta_n$

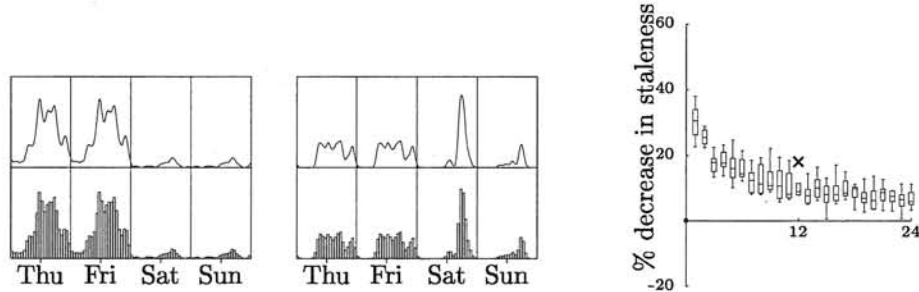

Figure 1: Left: part of the KDE developed for the business category together with a piecewise constant approximation. Middle: the same portion of the KDE for the FA Carling Premiership data. Right: percent decrease in staleness vs number of synchronisations per day for e-mail data.

and obtaining a distribution

$$p(\theta) = \frac{1}{N} \sum_{n=1}^{N} W\mathcal{N}(\theta|\theta_n, \sigma^2), \tag{7}$$

where $N$ is the number of data-points and the width parameters, $\sigma$, can be set through cross validation. Models of this type may be made use of for both $p_u(t)$ and $p_a(t)$.

### 3.2 Incorporating Prior Knowledge.

The underlying component frequencies of the data will clearly be more complex than simply a weekly or daily basis. Ideally we should be looking to incorporate as much of our prior knowledge about these component frequencies as possible. If we were modelling financial market's news, for example, we would expect weekdays to have similar characteristics to each other, but differing characteristics from the weekend. For this work, we considered four different scenarios of this type. For the first scenario, we took $T = 1$ day and placed no other constraints on the model. For the second we considered the longest period to be one week, $T = 1$ week, and placed no further constraints on the model. For the remaining two though we also considered $T$ to be one week, but we implemented further assumptions about the nature of the data. Firstly we split the data into weekdays and weekends. We then modelled these two categories separately, making sure that we maintained a continuous function for the whole week by wrapping basis functions between weekdays and weekends. Secondly we split the data into weekdays, Saturdays and Sundays, modelling each category separately and again wrapping basis functions across the days.

### 3.3 Model Selection.

To select the basis function widths, and to determine which periodicity assumption best matched the data, we utilised ten fold cross validation. For each different periodicity we used cross validation to first select the basis function width. We then compared the average likelihood across the ten validation sets, selecting the periodicity with the highest associated value.

## 4 Optimising the Synchronisation Times

Given that our user model, $p_a(t)$, and our data model, $p_u(t)$ will be a KDE based on wrapped Gaussians, we should be in a position to compute the required integrals

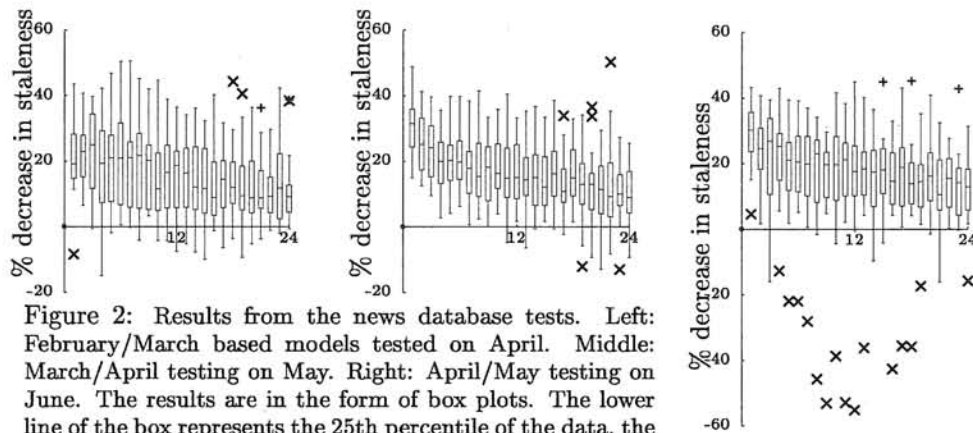

Figure 2: Results from the news database tests. Left: February/March based models tested on April. Middle: March/April testing on May. Right: April/May testing on June. The results are in the form of box plots. The lower line of the box represents the 25th percentile of the data, the upper line the 75th percentile and the central line the median. The 'whiskers' represent the maximum extent of the data up to $1.5 \times$ (75th percentile - 25th percentile). Data which lies outside the whiskers is marked with crosses.

in (5) and evaluate our objective function and derivatives thereof.

First though, we must give some attention to the target application for the algorithm. A known disadvantage of the standard kernel density estimate is the high storage requirements of the end model. The model requires that $N$ floating point numbers must be stored, where $N$ is the quantity of training data. Secondly, integrating across the cost function results in an objective function which is dependent on a large number of evaluations of the cumulative Gaussian distribution. Given that we envisage that such optimisations could be occurring within a PDA or mobile phone, it would seem prudent to seek a simpler approach to the required minimisation.

An alternative approach that we explored is to approximate the given distributions with a functional form which is more amenable to the integration. For example, a piecewise constant approximation to the KDE simplifies the integral considerably. It leads to a piecewise constant approximation for $\lambda_a(t)$ and $\lambda_u(t)$. Integration over which simply leads to a piecewise linear function which may be computed in a straightforward manner. Gradients may also be computed. We chose to reduce the optimisation to a series of one-dimensional line minimisations. This can be achieved in the following manner. First, note that the objective function, as a function of a particular synchronisation time $s_k$, may be written:

$$
\begin{aligned}
\langle C(s_k)\rangle \;=\; & \int_{s_{k-1}}^{s_k} \lambda_u(t)(s_k - t)\,dt + \int_{s_k}^{s_{k+1}} \lambda_u(s_{k+1} - t)\,dt \\
& + \int_{s_k}^{s_{k+1}} \lambda_a(t)(t - s_k)\,dt + \int_{s_{k-1}}^{s_k} \lambda_a(t - s_{k-1})\,dt + c(s_k) \quad (8)
\end{aligned}
$$

In other words, each synchronisation is only dependent on that of its neighbours. We may therefore perform the optimisation by visiting each synchronisation time, $s_k$, in a random order and optimising its position between its neighbours, which involves a one dimensional line minimisation of (8). This process, which is guaranteed to find a (local) minimum in our objective function, may be repeated until convergence.

# 5 Results

In this section we mainly explore the effectiveness of modelling the data-structures of interest. We will briefly touch upon the utility of modelling the cost evolution and user accesses in Section 5.2 but we leave a more detailed exploration of this area to later works.

## 5.1 Modelling Data Structures

To determine the effectiveness of our approach, we utilised two different sources of data: a news web-site and e-mail on a mail server.

The *news database data-set* was collected from the BBC News web site[4]. This site maintains a database of articles which are categorised according to subject, for example, UK News, Business News, Motorsport etc.. We had six months of data from February to July 2000 for 24 categories of the database.

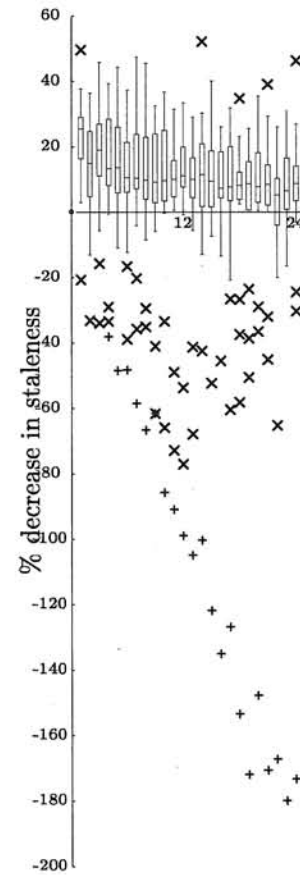

We modelled the data by decomposing it into the different categories and modelling each separately. This allowed us to explore the periodicity of each category independently. This is a sensible approach given that the nature of the data varies considerably across the categories. Two extreme examples are Business news and FA Carling Premiership news[5], Figure 1. Business news predominantly arrives during the week whereas FA Carling Premiership news arrives typically just after soccer games finish on a Saturday. Business news was best modelled on a Weekday/Weekend basis, and FA Carling Premiership news was best modelled on a Weekday/Saturday/Sunday basis. To evaluate the feasibility of our approach, we selected three consecutive months of data. The *inference step* consisted of constructing our models on data from the first two months. To restrict our investigations to the nature of the data evolution only, user access frequency was taken to be uniform and cost of connection was considered to be constant. For the *decision step* we considered 1 to 24 synchronisations a day. The synchronisation times were optimised for each category separately, they were initialised with a uniformly-spaced strategy, optimisation of the timings then proceeded as described in Section 4. The staleness associated with these timings was then computed for the third month. This value was compared with the staleness resulting from the uniformly-spaced strategy containing the same number of synchronisations[6]. The percentage decrease in staleness is shown in figures 2 and 3 in the form of box-plots.

Figure 3: May/June based models tested on July. + signifies the FA Carling Premiership Stream.

Generally an improvement in performance is observed, however, we note that in Figure 3 the performance for several categories is extremely

poor. In particular the FA Carling Premiership stream in Figure 3. The poor performance is caused by the soccer season ending in May. As a result relatively few articles are written in July, most of them concerning player transfer speculation, and the timing of those articles is very different from those in May. In other words the data evolves in a non-stationary manner which we have not modelled. The other poor performers are also sports related categories exhibiting non-stationarities.

The *e-mail data-set* was collected by examining the logs of e-mail arrival times for 9 researchers from Microsoft's Cambridge research lab. This data was collected for January and February 2001. We utilised the January data to build the probabilistic models and the February data to evaluate the average reduction in staleness. Figure 1 shows the results obtained.

In practice, a user is more likely to be interested in a *combination of different categories* of data. Perhaps several different streams of news and his e-mail. Therefore, to recreate a more realistic situation where a user has a combination of interests, we also collected e-mail arrivals for three users from February, March and April 2000. We randomly generated user profiles by sampling, without replacement, five categories from the available twenty-seven, rejecting samples where more than one e-mail stream was selected. We then modelled the users' interests by constructing an unweighted mixture of the five categories and proceeded to optimise the synchronisation times based on this model. This was performed one hundred times. The average staleness for the different numbers of synchronisations per day is shown in Figure 4.

Note that the performance for the combined categories is worse than it is for each individually. This is to be expected as the entropy of the combined model will always be greater than that of its constituents, we therefore have less information about arrival times, and as a result there are less gains to be made over the uniformly-spaced strategy[7].

## 5.2 Affect of Cost and User Model

In the previous sections we focussed on modelling the evolution of the databases. Here we now briefly turn our attention to the other portions of the system, user behaviour and connection cost. For this preliminary study, it proved difficult to obtain high quality data representing user access times. We therefore artificially generated a model which represents a user who accesses there device frequently at breakfast, lunchtime and during the evening, and rarely at night. Figure 4 simply shows the user model, $p_a(t)$, along with the result of optimising the cost function for uniform data arrivals and fixed cost under this user model. Note how synchronisation times are suggested just before high periods of user activity are about to occur. Also in Figure 4 is the effect of a varying cost, $c(t)$, under uniform $p_a(t)$ and $p_a(t)$.

Currently most mobile internet access providers appear to be charging a flat fee for call costs (typically in the U.K. about 15 cents per minute). However, when demand on their systems rise they may wish to incorporate a varying cost to flatten peak demands. This cost could be an actual cost for the user, or alternatively a 'shadow price' specified by service provider for controlling demand (Kelly, 2000). We give a simple example of such a call cost in Figure 4. For this we considered user access and data update rates to be constant. Note how the times move away from periods of high cost.

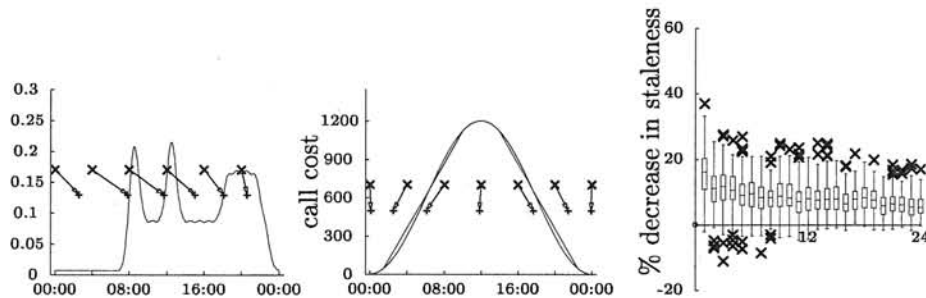

Figure 4: Left: change in synchronisation times for variable user access rates. × shows the initialisation points, + the end points. Middle: change in synchronisation times for a variable cost. Right: performance improvements for the combination of news and e-mail.

## 6 Discussion

The optimisation strategy we suggest could be sensitive to local minima, we did not try a range of different initialisations to explore this phenomena. However, by initialising with the uniformly-spaced strategy we ensured that we increased the objective function relative to the standard strategy. The month of July showed how a non-stationarity in the data structure can dramatically affect our performance. We are currently exploring on-line Bayesian models which we hope will track such non-stationarities.

The system we have explored in this work assumed that the data replicated on the mobile device was only modified on the server. A more general problem is that of *mutable replicas* where the data may be modified on the server or the client. Typical applications of such technology include mobile databases, where sales personnel modify portions of the database whilst on the road, and a calendar application on a PDA, where the user adds appointments on the PDA.

Finally there are many other applications of this type of technology beyond mobile devices. Web crawlers need to estimate when pages are modified to maintain a representative cache (Cho and Garcia-Molina, 2000) . Proxy servers could also be made to intelligent maintain their caches of web-pages up-to-date (Willis and Mikhailov, 1999; Wolman *et al.*, 1999) .

## Footnotes

[1]See, for example, AvantGo http://www.avantgo.com.

[2]When modelling user access times, if historic data is not available, models could also be constructed by querying the user about their likely activities.

[3]In practice we must approximate the wrapped distribution by restricting the number of terms in the sum.

[4]http://news.bbc.co.uk.

[5]The FA Carling Premiership is England's premier division soccer.

[6]The uniformly-spaced strategy's staleness varies with the timing of the first of the $K$ synchronisations. This figure was therefore an average of the staleness from all possible starting points taken at five minute intervals.

[7]The uniformly-spaced strategy can be shown to be optimal when the entropy of the underlying distribution is maximised (a uniform distribution across the interval).

## References

Cho, J. and H. Garcia-Molina (2000). Synchronizing a database to improve freshness. In *Proceedings 2000 ACM International Conference on Management of Data (SIG-MOD)*.

Kelly, F. P. (2000). Models for a self-managed internet. *Philosophical Transactions of the Royal Society* **A358**, 2335–2348.

Mardia, K. V. (1972). *Statistics of Directional Data*. London: Academic Press.

Rowstron, A. I. T., N. D. Lawrence, and C. M. Bishop (2001). Probabilistic modelling of replica divergence. In *Proceedings of the 8th Workshop on Hot Topics in Operating Systems HOTOS (VIII)*.

Willis, C. E. and M. Mikhailov (1999). Towards a better understanding of web resources and server responses for improved caching. In *Proceedings of the 8th International World Wide Web Conference*, pp. 153–165.

Wolman, A., G. M. Voelker, N. Sharma, N. Cardwell, A. Karlin, and H. M. Levy (1999). On the scale and performance of co-operative web proxy caching. In *17th ACM Symposium Operating System Principles (SOSP'99)*, pp. 16–31.

Yu, H. and A. Vahdat (2000). Design and evaluation of a continuous consistency model for replicated services. In *4th Symposium on Operating System Design and Implementation (OSDI)*.
